# REMARKS ON INTERPOLATION AND RECOGNITION USING NEURAL NETS

**Eduardo D. Sontag***
SYCON - Center for Systems and Control
Rutgers University
New Brunswick, NJ 08903

## Abstract

We consider different types of single-hidden-layer feedforward nets: with or without direct input to output connections, and using either threshold or sigmoidal activation functions. The main results show that direct connections in threshold nets double the recognition but not the interpolation power, while using sigmoids rather than thresholds allows (at least) doubling both. Various results are also given on VC dimension and other measures of recognition capabilities.

## 1  INTRODUCTION

In this work we continue to develop the theme of comparing threshold and sigmoidal feedforward nets. In (Sontag and Sussmann, 1989) we showed that the "generalized delta rule" (backpropagation) can give rise to pathological behavior –namely, the existence of spurious local minima even when no hidden neurons are used,– in contrast to the situation that holds for threshold nets. On the other hand, in (Sontag and Sussmann, 1989) we remarked that *provided that the right variant be used*, separable sets do give rise to globally convergent backpropagation, in complete analogy to the classical perceptron learning theorem. These results and those obtained by other authors probably settle most general questions about the case of no hidden units, so the next step is to look at the case of single hidden layers. In (Sontag, 1989) we announced the fact that sigmoidal activations (at least) double recognition power. Here we provide details, and we make several further remarks on this as well as on the topic of interpolation.

Nets with one hidden layer are known to be in principle sufficient for arbitrary recognition tasks. This follows from the approximation theorems proved by various

*E-mail: sontag@hilbert.rutgers.edu

authors: (Funahashi,1988), (Cybenko,1989), and (Hornik et. al., 1989). However, what is far less clear is *how many* neurons are needed for achieving a given recognition, interpolation, or approximation objective. This is of importance both in its practical aspects (having rough estimates of how many neurons will be needed is essential when applying backpropagation) and in evaluating generalization properties (larger nets tend to lead to poorer generalization). It is known and easy to prove (see for instance (Arai, 1989), (Chester, 1990)) that one can basically interpolate values at any $n + 1$ points using an $n$-neuron net, and in particular that any $n + 1$-point set can be dichotomized by such nets. Among other facts, we point out here that allowing direct input to output connections permits doubling the recognition power to $2n$, and the same result is achieved if sigmoidal neurons are used but such direct connections are not allowed. Further, we remark that approximate interpolation of $2n - 1$ points is also possible, provided that sigmoidal units be employed (but direct connections in threshold nets do not suffice).

The dimension of the input space (that is, the number of "input units") can influence the number of neurons needed, are least for dichotomy problems for suitably chosen sets. In particular, Baum had shown some time back (Baum, 1988) that the VC dimension of threshold nets with a fixed number of hidden units is at least proportional to this dimension. We give lower bounds, in dimension two, at least doubling the VC dimension if sigmoids or direct connections are allowed.

Lack of space precludes the inclusion of proofs; references to technical reports are given as appropriate. A full-length version of this paper is also available from the author.

## 2   DICHOTOMIES

The first few definitions are standard. Let $N$ be a positive integer. A *dichotomy* or *two-coloring* $(S_-, S_+)$ on a set $S \subseteq \mathbb{R}^N$ is a partition $S = S_- \bigcup S_+$ of $S$ into two disjoint subsets. A function $f : \mathbb{R}^N \to \mathbb{R}$ will be said to *implement* this dichotomy if it holds that

$$f(u) > 0 \text{ for } u \in S_+ \text{ and } f(u) < 0 \text{ for } u \in S_- \ .$$

Let $\mathcal{F}$ be a class of functions from $\mathbb{R}^N$ to $\mathbb{R}$, assumed to be nontrivial, in the sense that for each point $u \in \mathbb{R}^N$ there is some $f_1 \in \mathcal{F}$ so that $f_1(u) > 0$ and some $f_2 \in \mathcal{F}$ so that $f_2(u) < 0$. This class *shatters* the set $S \subseteq R^N$ if each dichotomy on $S$ can be implemented by some $f \in \mathcal{F}$.

Here we consider, for any class of functions $\mathcal{F}$ as above, the following measures of classification power. First we introduce $\overline{\mu}$ and $\underline{\mu}$, dealing with "best" and "worst" cases respectively: $\overline{\mu}(\mathcal{F})$ denotes the largest integer $l \geq 1$ (possibly $\infty$) so that there is at least *some* set $S$ of cardinality $l$ in $\mathbb{R}^N$ which can be shattered by $\mathcal{F}$, while $\underline{\mu}(\mathcal{F})$ is the largest integer $l \geq 1$ (possibly $\infty$) so that *every* set of cardinality $l$ can be shattered by $\mathcal{F}$. Note that by definition, $\underline{\mu}(\mathcal{F}) \leq \overline{\mu}(\mathcal{F})$ for every class $\mathcal{F}$.

In particular, the definitions imply that no set of cardinality $\overline{\mu}(\mathcal{F}) + 1$ can be shattered, and that there is at least some set of cardinality $\underline{\mu}(\mathcal{F}) + 1$ which cannot be shattered. The integer $\overline{\mu}$ is usually called the *Vapnik-Chervonenkis (VC) dimension* of the class $\mathcal{F}$ (see for instance (Baum,1988)), and appears in formalizations of learning in the distribution-free sense.

A set may fail to be shattered by $\mathcal{F}$ because it is very special (see the example below with colinear points). In that sense, a more robust measure is useful: $\mu(\mathcal{F})$ is the largest integer $l \geq 1$ (possibly $\infty$) for which the class of sets $S$ that can be shattered by $\mathcal{F}$ is dense, in the sense that given every $l$-element set $S = \{s_1, \ldots, s_l\}$ there are points $\widetilde{s}_i$ arbitrarily close to the respective $s_i$'s such that $\widetilde{S} = \{\widetilde{s}_1, \ldots, \widetilde{s}_l\}$ can be shattered by $\mathcal{F}$. Note that

$$\underline{\mu}(\mathcal{F}) \leq \mu(\mathcal{F}) \leq \overline{\mu}(\mathcal{F}) \tag{1}$$

for all $\mathcal{F}$.

To obtain an upper bound $m$ for $\mu(\mathcal{F})$ one needs to exhibit an open class of sets of cardinality $m + 1$ none of which can be shattered.

Take as an example the class $\mathcal{F}$ consisting of all affine functions $f(x) = ax + by + c$ on $\mathbb{R}^2$. Since any three points can be shattered by an affine map provided that they are not colinear (just choose a line $ax + by + c = 0$ that separates any point which is colored different from the rest), it follows that $3 \leq \mu$. On the other hand, no set of four points can ever be dichotomized, which implies that $\overline{\mu} \leq 3$ and therefore the conclusion $\mu = \overline{\mu} = 3$ for this class. (The negative statement can be verified by a case by case analysis: if the four points form the vertices of a 4-gon color them in "XOR" fashion, alternate vertices of the same color; if 3 form a triangle and the remaining one is inside, color the extreme points differently from the remaining one; if all colinear then use an alternating coloring). Finally, since there is some set of 3 points which cannot be dichotomized (any set of three colinear points is like this), but every set of two can, $\underline{\mu} = 2$.

We shall say that $\mathcal{F}$ is *robust* if whenever $S$ can be shattered by $\mathcal{F}$ also every small enough perturbation of $S$ can be shattered. For a robust class and $l = \mu(\mathcal{F})$, every set in an open dense subset in the above topology, i.e. *almost every* set of $l$ elements, can be shattered.

## 3   NETS

We define a "neural net" as a function of a certain type, corresponding to the idea of feedforward interconnections, via additive links, of neurons each of which has a scalar response or *activation function* $\theta$.

**Definition 3.1** Let $\theta : \mathbb{R} \to \mathbb{R}$ be any function. A function $f : \mathbb{R}^N \to \mathbb{R}$ is a *single-hidden-layer neural net with $k$ hidden neurons of type $\theta$ and $N$ inputs*, or just a *$(k, \theta)$-net*, if there are real numbers $w_0, w_1, \ldots, w_k, \tau_1, \ldots, \tau_k$ and vectors $v_0, v_1, \ldots, v_k \in \mathbb{R}^N$ such that, for all $u \in \mathbb{R}^N$,

$$f(u) = w_0 + v_0.u + \sum_{i=1}^{k} w_i \theta(v_i.u - \tau_i) \tag{2}$$

where the dot indicates inner product. *A net with no direct i/o connections is one for which $v_0 = 0$.*

For fixed $\theta$, and under mild assumptions on $\theta$, such neural nets can be used to approximate uniformly arbitrary continuous functions on compacts. In particular, they can be used to implement arbitrary dichotomies.

In neural net practice, one often takes $\theta$ to be the *standard sigmoid* $\sigma(x) = \frac{1}{1+e^{-x}}$ or equivalently, up to translations and change of coordinates, the hyperbolic tangent $\tanh(x)$. Another usual choice is the hardlimiter, threshold, or *Heaviside* function

$$\mathcal{H}(x) = \begin{cases} 0 & \text{if } x \leq 0 \\ 1 & \text{if } x > 0 \end{cases}$$

which can be approximated well by $\sigma(\gamma x)$ when the "gain" $\gamma$ is large. Yet another possibility is the use of the piecewise linear function

$$\pi(x) = \begin{cases} -1 & \text{if } x \leq -1 \\ 1 & \text{if } x \geq 1 \\ x & \text{otherwise.} \end{cases}$$

Most analysis has been done for $\mathcal{H}$ and no direct connections, but numerical techniques typically use the standard sigmoid (or equivalently tanh). The activation $\pi$ will be useful as an example for which sharper bounds can be obtained. The examples $\sigma$ and $\pi$, but not $\mathcal{H}$, are particular cases of the following more general type of activation function:

**Definition 3.2** A function $\theta : \mathbb{R} \to \mathbb{R}$ will be called a *sigmoid* if these two properties hold:

(S1) $t_+ := \lim_{x \to +\infty} \theta(x)$ and $t_- := \lim_{x \to -\infty} \theta(x)$ exist, and $t_+ \neq t_-$.

(S2) There is some point $c$ such that $\theta$ is differentiable at $c$ and $\theta'(c) = \mu \neq 0$.    □

All the examples above lead to robust classes, in the sense defined earlier. More precisely, assume that $\theta$ is continuous except for at most finitely many points $x$, and it is left continuous at such $x$, and let $\mathcal{F}$ be the class of $(k, \theta)$-nets, for any fixed $k$. Then $\mathcal{F}$ is robust, and the same statement holds for nets with no direct connections.

## 4    CLASSIFICATION RESULTS

We let $\mu(k, \theta, N)$ denote $\mu(\mathcal{F})$, where $\mathcal{F}$ is the class of $(k, \theta)$-nets in $\mathbb{R}^N$ with *no direct connections*, and similarly for $\underline{\mu}$ and $\overline{\mu}$, and a superscript $d$ is used for the class of arbitrary such nets (with possible direct connections from input to output). The lower measure $\underline{\mu}$ is independent of dimension:

**Lemma 4.1** For each $k, \theta, N$, $\underline{\mu}(k, \theta, N) = \underline{\mu}(k, \theta, 1)$ and $\underline{\mu}^d(k, \theta, N) = \underline{\mu}^d(k, \theta, 1)$.

This justifies denoting these quantities just as $\underline{\mu}(k, \theta)$ and $\underline{\mu}^d(k, \theta)$ respectively, as we do from now on, and giving proofs only for $N = 1$.

**Lemma 4.2** For any sigmoid $\theta$, and for each $k, N$,

$$\mu(k + 1, \theta, N) \geq \mu^d(k, \mathcal{H}, N)$$

and similarly for $\underline{\mu}$ and $\overline{\mu}$.

The main results on classification will be as follows.

**Theorem 1** *For any sigmoid $\theta$, and for each $k$,*

$$\underline{\mu}(k,\mathcal{H}) = k+1$$
$$\underline{\mu}^d(k,\mathcal{H}) = 2k+2$$
$$\underline{\mu}(k,\theta) \geq 2k .$$

**Theorem 2** *For each $k$,*

$$4\left\lfloor\frac{k}{2}\right\rfloor \leq \mu(k,\mathcal{H},2) \leq 2k+1$$
$$\mu^d(k,\mathcal{H},2) \leq 4k+3 .$$

**Theorem 3** *For any sigmoid $\theta$, and for each $k$,*

$$2k+1 \leq \overline{\mu}(k,\mathcal{H},2)$$
$$4k+3 \leq \overline{\mu}^d(k,\mathcal{H},2)$$
$$4k-1 \leq \overline{\mu}(k,\theta,2) .$$

These results are proved in (Sontag, 1990a). The first inequality in Theorem 2 follows from the results in (Baum, 1988), who in fact established a lower bound of $2N\lfloor\frac{k}{2}\rfloor$ for $\mu(k,\mathcal{H},N)$ (and hence for $\overline{\mu}$ too), for every $N$, not just $N = 2$ as in the Theorem above. We conjecture, but have as yet been unable to prove, that direct connections or sigmoids should also improve these bounds by at least a factor of 2, just as in the two-dimensional case and in the worst-case analysis. Because of Lemma 4.2, the last statements in Theorems 1 and 3 are consequences of the previous two.

## 5   SOME PARTICULAR ACTIVATION FUNCTIONS

Consider the last inequality in Theorem 1. For arbitrary sigmoids, this is far too conservative, as the number $\mu$ can be improved considerably from $2k$, even made infinite (see below). We conjecture that for the important practical case $\theta(x) = \sigma(x)$ it is close to optimal, but the only upper bounds that we have are still too high. For the piecewise linear function $\pi$, at least, one has equality:

**Lemma 5.1** $\underline{\mu}(k,\pi) = 2k$.

It is worth remarking that there are sigmoids $\theta$, as differentiable as wanted, even real-analytic, where all classification measures are infinite. Of course, the function $\theta$ is so complicated that there is no reasonably "finite" implementation for it. This remark is only of theoretical interest, to indicate that, unless further restrictions are made on (S1)-(S2), much better bounds can be obtained. (If only $\mu$ and $\overline{\mu}$ are desired to be infinite, one may also take the simpler example $\theta(x) = \sin(x)$. Note that for any $l$ rationally independent real numbers $x_i$, the vectors of the form $(\sin(\gamma_1 x_1),\ldots,\sin(\gamma_l x_l)$, with the $\gamma_i$'s real, form a dense subset of $[-1,1]^l$, so all dichotomies on $\{x_1,\ldots,x_l\}$ can be implemented with $(1,\sin)$-nets.)

**Lemma 5.2** There is some sigmoid $\theta$, which can be taken to be an analytic function, so that $\underline{\mu}(1,\theta) = \infty$.

## 6  INTERPOLATION

We now consider the following approximate interpolation problem. Assume given a sequence of $k$ (distinct) points $x_1, \ldots, x_k$ in $R^N$, any $\varepsilon > 0$, and any sequence of real numbers $y_1, \ldots, y_k$, as well as some class $\mathcal{F}$ of functions from $\mathbb{R}^N$ to $\mathbb{R}$. We ask if there exists some

$$f \in \mathcal{F} \text{ so that } |f(x_i) - y_i| < \varepsilon \text{ for each } i . \tag{3}$$

Let $\underline{\lambda}(\mathcal{F})$ be the largest integer $k \geq 1$, possibly infinite, so that for every set of data as above (3) can be solved. Note that, obviously, $\underline{\lambda}(\mathcal{F}) \leq \underline{\mu}(\mathcal{F})$. Just as in Lemma 4.1, $\underline{\lambda}$ is independent of the dimension $N$ when applied to nets. Thus we let $\underline{\lambda}^d(k, \theta)$ and $\underline{\lambda}(k, \theta)$ be respectively the values of $\underline{\lambda}(\mathcal{F})$ when applied to $(k, \theta)$-nets with or without direct connections.

We now summarize properties of $\underline{\lambda}$. The next result —see (Sontag,1991), as well as the full version of this paper, for a proof— should be compared with Theorem 1. The main difference is in the second equality. Note that one can prove $\underline{\lambda}(k, \theta) \geq \underline{\lambda}^d(k - 1, \mathcal{H})$, in complete analogy with the case of $\underline{\mu}$, but this is not sufficient anymore to be able to derive the last inequality in the Theorem from the second equality.

**Theorem 4** *For any continuous sigmoid $\theta$, and for each $k$,*

$$\underline{\lambda}(k, \mathcal{H}) = k + 1$$
$$\underline{\lambda}^d(k, \mathcal{H}) = k + 2$$
$$\underline{\lambda}(k, \theta) \geq 2k - 1 .$$

**Remark 6.1** Thus we can approximately interpolate any $2k - 1$ points using $k$ sigmoidal neurons. It is not hard to prove as a corollary that, for the standard sigmoid, this approximate interpolation property holds in the following stronger sense: for an open dense set of $2k - 1$ points, one can achieve an open dense set of values; the proof involves looking first at points with rational coordinates, and using that on such points one is dealing basically with rational functions (after a diffeomorphism), plus some theory of semialgebraic sets. We conjecture that one should be able to interpolate at $2k$ points. Note that for $k = 2$ this is easy to achieve: just choose the slope $d$ so that some $z_i - z_{i+1}$ becomes zero and the $z_i$ are allowed to be nonincreasing or nondecreasing. The same proof, changing the signs if necessary, gives the wanted net. For some examples, it is quite easy to get $2k$ points. For instance, $\underline{\lambda}(k, \pi) = 2k$ for the piecewise linear sigmoid $\pi$.    □

## 7  FURTHER REMARKS

The main conclusion from Theorem 1 is that sigmoids at least double recognition power for arbitrary sets. It may be the case that $\overline{\mu}(k, \sigma, N)/\overline{\mu}(k, \mathcal{H}, N) \approx 2$ for all $N$; this is true for $N = 1$ and is strongly suggested by Theorem 3 (the first bound appears to be quite tight). Unfortunately the proof of this theorem is based on a result from (Asano et. al., 1990) regarding arrangements of points in the plane, a fact which does not generalize to dimension three or higher.

One may also compare the power of nets with and without connections, or threshold vs sigmoidal processors, on Boolean problems. For instance, it is a trivial consequence from the given results that parity on $n$ bits can be computed with $\lceil \frac{n+1}{2} \rceil$

hidden sigmoidal units and no direct connections, though requiring (apparently, though this is an open problem) $n$ thresholds. In addition, for some families of Boolean functions, the gap between sigmoidal nets and threshols nets may be infinitely large (Sontag, 1990a). See (Sontag, 1990b) for representation properties of *two*-hidden-layer nets

## Acknowledgements

This work was supported in part by Siemens Corporate Research, and in part by the CAIP Center, Rutgers University.

## References

Arai, M., "Mapping abilities of three-layer neural networks," *Proc. IJCNN Int.Joint Conf.on Neural Networks, Washington, June 18-22, 1989*, IEEE Publications, 1989, pp. I-419/424.

Asano,T., J. Hershberger, J. Pach, E.D. Sontag, D. Souivaine, and S. Suri, "Separating Bi-Chromatic Points by Parallel Lines," *Proceedings of the Second Canadian Conference on Computational Geometry*, Ottawa, Canada, 1990, p. 46-49.

Baum, E.B., "On the capabilities of multilayer perceptrons," *J.Complexity* 4(1988): 193-215.

Chester, D., "Why two hidden layers and better than one," *Proc. Int. Joint Conf. on Neural Networks*, Washington, DC, Jan. 1990, IEEE Publications, 1990, p. I.265-268.

Cybenko, G., "Approximation by superpositions of a sigmoidal function," *Math. Control, Signals, and Systems* 2(1989): 303-314.

Funahashi, K., "On the approximate realization of continuous mappings by neural networks," *Proc. Int. Joint Conf. on Neural Networks*, IEEE Publications, 1988, p. I.641-648.

Hornik, K.M., M. Stinchcombe, and H. White, "Multilayer feedforward networks are universal approximators," *Neural Networks* 2(1989): 359-366.

Sontag, E.D., "Sigmoids distinguish better than Heavisides," *Neural Computation* 1(1989): 470-472.

Sontag, E.D., "On the recognition capabilities of feedforward nets," Report SYCON-90-03, *Rutgers Center for Systems and Control*, April 1990.

Sontag, E.D., "Feedback Stabilization Using Two-Hidden-Layer Nets," Report SYCON-90-11, *Rutgers Center for Systems and Control*, October 1990.

Sontag, E.D., "Capabilities and training of feedforward nets," in *Theory and Applications of Neural Networks* (R. Mammone and J. Zeevi, eds.), Academic Press, NY, 1991, to appear.

Sontag, E.D., and H.J. Sussmann, "Backpropagation can give rise to spurious local minima even for networks without hidden layers," *Complex Systems* 3(1989): 91-106.

Sontag, E.D., and H.J. Sussmann, "Backpropagation separates where perceptrons do," *Neural Networks*(1991), to appear.
